# Cost-Sensitive Exploration in Bayesian Reinforcement Learning

**Dongho Kim**
Department of Engineering
University of Cambridge, UK
dk449@cam.ac.uk

**Kee-Eung Kim**
Dept of Computer Science
KAIST, Korea
kekim@cs.kaist.ac.kr

**Pascal Poupart**
School of Computer Science
University of Waterloo, Canada
ppoupart@cs.uwaterloo.ca

## Abstract

In this paper, we consider Bayesian reinforcement learning (BRL) where actions incur costs in addition to rewards, and thus exploration has to be constrained in terms of the expected total cost while learning to maximize the expected long-term total reward. In order to formalize cost-sensitive exploration, we use the constrained Markov decision process (CMDP) as the model of the environment, in which we can naturally encode exploration requirements using the cost function. We extend BEETLE, a model-based BRL method, for learning in the environment with cost constraints. We demonstrate the cost-sensitive exploration behaviour in a number of simulated problems.

## 1  Introduction

In reinforcement learning (RL), the agent interacts with a (partially) unknown environment, classically assumed to be a Markov decision process (MDP), with the goal of maximizing its expected long-term total reward. The agent faces the exploration-exploitation dilemma: the agent must select actions that exploit its current knowledge about the environment to maximize reward, but it also needs to select actions that explore for more information so that it can act better. Bayesian RL (BRL) [1, 2, 3, 4] provides a principled framework to the exploration-exploitation dilemma.

However, exploratory actions may have serious consequences. For example, a robot exploring in an unfamiliar terrain may reach a dangerous location and sustain heavy damage, or wander off from the recharging station to the point where a costly rescue mission is required. In a less mission critical scenario, a route recommendation system that learns actual travel times should be aware of toll fees associated with different routes. Therefore, the agent needs to carefully (if not completely) avoid critical situations while exploring to gain more information.

The constrained MDP (CMDP) extends the standard MDP to account for limited resources or multiple objectives [5]. The CMDP assumes that executing actions incur costs *and* rewards that should be optimized separately. Assuming the expected total reward and cost criterion, the goal is to find an optimal policy that maximizes the expected total reward while bounding the expected total cost. Since we can naturally encode undesirable behaviors into the cost function, we formulate the cost-sensitive exploration problem as RL in the environment modeled as a CMDP.

Note that we can employ other criteria for the cost constraint in CMDPs. We can make the actual total cost below the cost bound with probability one using the *sample-path cost* constraints [6, 7], or with probability $1 - \delta$ using the *percentile cost* constraints [8]. In this paper, we restrict ourselves to the expected total cost constraint mainly due to the computational efficiency in solving the constrained optimization problem. Extending our work to other cost criteria is left as a future work. The main argument we make is that the CMDP provides a natural framework for representing various approaches to constrained exploration, such as *safe exploration* [9, 10].

In order to perform cost-sensitive exploration in the Bayesian RL (BRL) setting, we cast the problem as a constrained partially observable MDP (CPOMDP) [11, 12] planning problem. Specifically, we take a model-based BRL approach and extend BEETLE [4] to solve the CPOMDP which models BRL with cost constraints.

## 2   Background

In this section, we review the background for cost-sensitive exploration in BRL. As we explained in the previous section, we assume that the environment is modeled as a CMDP, and formulate model-based BRL as a CPOMDP. We briefly review the CMDP and CPOMDP before summarizing BEETLE, a model-based BRL for environments without cost constraints.

### 2.1   Constrained MDPs (CMDPs) and Constrained POMDPs (CPOMDPs)

The standard (infinite-horizon discounted return) MDP is defined by tuple $\langle S, A, T, R, \gamma, b_0 \rangle$ where: $S$ is the set of states $s$; $A$ is the set of actions $a$; $T(s, a, s')$ is the transition function which denotes the probability $\Pr(s'|s, a)$ of changing to state $s'$ from $s$ by executing action $a$; $R(s, a) \in \Re$ is the reward function which denotes the immediate reward of executing action $a$ in state $s$; $\gamma \in [0, 1)$ is the discount factor; $b_0(s)$ is the initial state probability for state $s$. $b_0$ is optional, since an optimal policy $\pi^* : S \rightarrow A$ that maps from states to actions can be shown not to be dependent on $b_0$.

The constrained MDP (CMDP) is defined by tuple $\langle S, A, T, R, C, \hat{c}, \gamma, b_0 \rangle$ with the following additional components: $C(s, a) \in \Re$ is the cost function which denotes the immediate cost incurred by executing action $a$ in state $s$; $\hat{c}$ is the bound on expected total discounted cost.

An optimal policy of a CMDP maximizes the expected total discounted reward over the infinite horizon, while not incurring more than $\hat{c}$ total discounted cost in the expectation. We can formalize this constrained optimization problem as:

$$\max_\pi \mathcal{V}^\pi \quad \text{s.t.} \quad \mathcal{C}^\pi \leq \hat{c}.$$

where $\mathcal{V}^\pi = E_{\pi, b_0}[\sum_{t=0}^\infty \gamma^t R(s_t, a_t)]$ is the expected total discounted reward, and $\mathcal{C}^\pi = E_{\pi, b_0}[\sum_{t=0}^\infty \gamma^t C(s_t, a_t)]$ is the expected total discounted cost. We will also use $\mathcal{C}^\pi(s)$ to denote the expected total cost starting from the state $s$.

It has been shown that an optimal policy for CMDP is generally a randomized stationary policy [5]. Hence, we define a policy $\pi$ as a mapping of states to probability distributions over actions, where $\pi(s, a)$ denotes the probability that an agent will execute action $a$ in state $s$. We can find an optimal policy by solving the following linear program (LP):

$$\max_x \sum_{s,a} R(s, a) x(s, a) \tag{1}$$

$$\text{s.t.} \ \sum_a x(s', a) - \gamma \sum_{s,a} x(s, a) T(s, a, s') = b_0(s') \quad \forall s'$$

$$\sum_{s,a} C(s, a) x(s, a) \leq \hat{c} \ \text{ and } \ x(s, a) \geq 0 \quad \forall s, a$$

The variables $x$'s are related to the occupancy measure of optimal policy, where $x(s, a)$ is the expected discounted number of times executing $a$ at state $s$. If the above LP yields a feasible solution, optimal policy can be obtained by $\pi(s, a) = x(s, a) / \sum_{a'} x(s, a')$. Note that due to the introduction of cost constraints, the resulting optimal policy is contingent on the initial state distribution $b_0$, in contrast to the standard MDP of which an optimal policy can be independent of the initial state distribution. Note also that the above LP may be infeasible if there is no policy that can satisfy the cost constraint.

The constrained POMDP (CPOMDP) extends the standard POMDP in a similar manner. The standard POMDP is defined by tuple $\langle S, A, Z, T, O, R, \gamma, b_0 \rangle$ with the following additional components: the set $Z$ of observations $z$, and the observation probability $O(s', a, z)$ representing the probability $\Pr(z|s', a)$ of observing $z$ when executing action $a$ and changing to state $s'$. The states in the POMDP are hidden to the agent, and it has to act based on the observations instead. The CPOMDP

---

**Algorithm 1:** Point-based backup of $\alpha$-vector pairs with admissible cost

---

**input** : $(b, d)$ with belief state $b$ and admissible cost $d$; set $\Gamma$ of $\alpha$-vector pairs
**output**: set $\Gamma'_{(b,d)}$ of $\alpha$-vector pairs (contains at most 2 pairs for a single cost function)

// regress
**foreach** $a \in A$ **do**
     $\alpha_R^{a,*} = R(\cdot, a), \ \alpha_C^{a,*} = C(\cdot, a)$
     **foreach** $(\alpha_{i,R}, \alpha_{i,C}) \in \Gamma, z \in Z$ **do**
         $\alpha_{i,R}^{a,z}(s) = \sum_{s'} T(s, a, s') O(s', a, z) \alpha_{i,R}(s')$
         $\alpha_{i,C}^{a,z}(s) = \sum_{s'} T(s, a, s') O(s', a, z) \alpha_{i,C}(s')$

// backup for each action
**foreach** $a \in A$ **do**
     Solve the following LP to obtain best randomized action at the next time step:

$$\max_{\tilde{w}_{iz}, d_z} b \cdot \sum_{i,z} \tilde{w}_{iz} \alpha_{i,R}^{a,z} \quad \text{subject to} \quad \begin{aligned} & b \cdot \sum_i \tilde{w}_{iz} \alpha_{i,C}^{a,z} \leq d_z \quad \forall z \\ & \sum_i \tilde{w}_{iz} = 1 \quad \forall z \\ & \tilde{w}_{iz} \geq 0 \quad \forall i, z \\ & \sum_z d_z = \tfrac{1}{\gamma}(d - C(b,a)) \end{aligned}$$

     $\alpha_R^a = \alpha_R^{a,*} + \gamma \sum_{i,z} \tilde{w}_{iz} \alpha_{i,R}^{a,z}$
     $\alpha_C^a = \alpha_C^{a,*} + \gamma \sum_{i,z} \tilde{w}_{iz} \alpha_{i,C}^{a,z}$

// find the best randomized action for the current time step
Solve the following LP with :

$$\max_{w_a} b \cdot \sum_a w_a \alpha_R^a \quad \text{subject to} \quad \begin{aligned} & b \cdot \sum_a w_a \alpha_C^a \leq d \\ & \sum_a w_a = 1 \\ & w_a \geq 0 \quad \forall a \end{aligned}$$

**return** $\Gamma'_{(b,d)} = \{(\alpha_R^a, \alpha_C^a) | w_a > 0\}$

---

is defined by adding the cost function $C$ and the cost bound $\hat{c}$ into the definition as in the CMDP. Although the CPOMDP is intractable to solve as is the case with the POMDP, there exists an efficient point-based algorithm [12].

The Bellman backup operator for CPOMDP generates pairs of $\alpha$-vectors $(\alpha_R, \alpha_C)$, each vector corresponding to the expected total reward and cost, respectively. In order to facilitate defining the Bellman backup operator at a belief state, we augment the belief state with a scalar quantity called *admissible cost* [13], which represents the expected total cost that can be additionally incurred for the future time steps without violating the cost constraint. Suppose that, at time step $t$, the agent has so far incurred a total cost of $W_t$, i.e., $W_t = \sum_{\tau=0}^{t} \gamma^\tau C(s_\tau, a_\tau)$. The admissible cost at time step $t + 1$ is defined as $d_t = \frac{1}{\gamma^{t+1}}(\hat{c} - W_t)$. It can be computed recursively by the equation $d_{t+1} = \frac{1}{\gamma}(d_t - C(s_t, a_t))$, which can be derived from $W_t = W_{t-1} + \gamma C(s_t, a_t)$, and $d_0 = \hat{c}$. Given a pair of belief state and admissible cost $(b, d)$ and the set of $\alpha$-vector pairs $\Gamma = \{(\alpha_{i,R}, \alpha_{i,C})\}$, the best (randomized) action is obtained by solving the following LP:

$$\max_{w_i} b \cdot \sum_i w_i \alpha_{i,R} \quad \text{subject to} \quad \begin{aligned} & b \cdot \sum_i w_i \alpha_{i,C} \leq d \\ & \sum_i w_i = 1 \\ & w_i \geq 0 \quad \forall i \end{aligned}$$

where $w_i$ corresponds to the probability of choosing the action associated with the pair $(\alpha_{i,R}, \alpha_{i,C})$. The point-based backup for CPOMDP leveraging the above LP formulation is shown in Algorithm 1.[1]

## 2.2 BEETLE

BEETLE [4] is a model-based BRL algorithm, based on the idea that BRL can be formulated as a POMDP planning problem. Assuming that the environment is modeled as a discrete-state MDP $P = \langle S, A, T, R, \gamma \rangle$ where the transition function $T$ is unknown, we treat each transition probability $T(s, a, s')$ as an unknown parameter $\theta_a^{s,s'}$ and formulate BRL as a hyperstate POMDP $\langle S_P, A_P, Z_P, T_P, O_P, R_P, \gamma, b_0 \rangle$ where $S_P = S \times \{\theta_a^{s,s'}\}$, $A_P = A$, $Z_P = S$, $T_P(s, \theta, a, s', \theta') = \theta_a^{s,s'} \delta_\theta(\theta')$, $O_P(s', \theta', a, z) = \delta_{s'}(z)$, and $R_P(s, \theta, a) = R(s, a)$. In summary, the hyperstate POMDP augments the original state space with the set of unknown parameters $\{\theta_a^{s,s'}\}$, since the agent has to take actions without exact information on the unknown parameters.

The belief state $b$ in the hyperstate POMDP yields the posterior of $\theta$. Specifically, assuming a product of Dirichlets for the belief state such that

$$b(\theta) = \prod_{s,a} \mathrm{Dir}(\theta_a^{s,*}; n_a^{s,*})$$

where $\theta_a^{s,*}$ is the parameter vector of multinomial distribution defining the transition function for state $s$ and action $a$, and $n_a^{s,*}$ is the hyperparameter vector of the corresponding Dirichlet distribution. Since the hyperparameter $n_a^{s,s'}$ can be viewed as *pseudocounts*, i.e., the number of times observing transition $(s, a, s')$, the updated belief after observing transition $(\hat{s}, \hat{a}, \hat{s}')$ is also a product of Dirichlets:

$$b_{\hat{a}}^{\hat{s}, \hat{s}'}(\theta) = \prod_{s,a} \mathrm{Dir}(\theta_a^{s,*}; n_a^{s,*} + \delta_{\hat{s}, \hat{a}, \hat{s}'}(s, a, s'))$$

Hence, belief states in the hyperstate POMDP can be represented by $|S|^2 |A|$ variables one for each hyperparameter, and the belief update is efficiently performed by incrementing the hyperparmeter corresponding to the observed transition.

Solving the hyperstate POMDP is performed by dynamic programming with the Bellman backup operator [2]. Specifically, the value function is represented as a set $\Gamma$ of $\alpha$-functions for each state $s$, so that the value of optimal policy is obtained by $\mathcal{V}_s^*(b) = \max_{\alpha \in \Gamma} \alpha_s(b)$ where $\alpha_s(b) = \int_\theta b(\theta) \alpha_s(\theta) d\theta$. Using the fact that $\alpha$-functions are multivariate polynomials of $\theta$, we can obtain an exact solution to the Bellman backup.

There are two computational challenges with the hyperstate POMDP approach. First, being a POMDP, the Bellman backup has to be performed on all possible belief states in the probability simplex. BEETLE adopts Perseus [14], performing randomized point-based backups confined to the set of sampled $(s, b)$ pairs by simulating a default or random policy, and reducing the total number of value backups by improving the value of many belief points through a single backup. Second, the number of monomial terms in the $\alpha$-function increases exponentially with the number of backups. BEETLE chooses a fixed set of basis functions and projects the $\alpha$-function onto a linear combination of these basis functions. The set of basis functions is chosen to be the set of monomials extracted from the sampled belief states.

## 3 Constrained BEETLE (CBEETLE)

We take an approach similar to BEETLE for cost-sensitive exploration in BRL. Specifically, we formulate cost-sensitive BRL as a hyperstate CPOMDP $\langle S_P, A_P, Z_P, T_P, O_P, R_P, C_P, \hat{c}, \gamma, b_0 \rangle$ where $S_P = S \times \{\theta_a^{s,s'}\}$, $A_P = A$, $Z_P = S$, $T_P(s, \theta, a, s', \theta') = \theta_a^{s,s'} \delta_\theta(\theta')$, $O_P(s', \theta', a, z) = \delta_{s'}(z)$, $R_P(s, \theta, a) = R(s, a)$, and $C_P(s, \theta, a) = C(s, a)$.

Note that using the cost function $C$ and cost bound $\hat{c}$ to encode the constraints on the exploration behaviour allows us to enjoy the same flexibility as using the reward function to define the task objective in the standard MDP and POMDP. Although, for the sake of exposition, we use a single cost function and discount factor in our definition of CMDP and CPOMDP, we can generalize the model to have multiple cost functions that capture different aspects of exploration behaviour that cannot be put together on the same scale, and different discount factors for rewards and costs. In addition, we can even completely eliminate the possibility of executing action $a$ in state $s$ by setting the discount factor to 1 for the cost constraint and impose a sufficiently low cost bound $\hat{c} < C(s, a)$.

---

**Algorithm 2:** Point-based backup of $\alpha$-function pairs for the hyperstate CPOMDP[2]

---

**input** : $(s, n, d)$ with state $s$, Dirichlet hyperparameter $n$ representing belief state $b$, and admissible cost $d$; set $\Gamma_s$ of $\alpha$-function pairs for each state $s$

**output**: set $\Gamma'_{(s,n,d)}$ of $\alpha$-function pairs (contains at most 2 pairs for a single cost function)

// regress

**foreach** $a \in A$ **do**

    $\alpha_R^{a,*} = R(s,a)$, $\alpha_C^{a,*} = C(s,a)$    // constant functions

    **foreach** $s' \in S, (\alpha_{i,R}, \alpha_{i,C}) \in \Gamma_{s'}$ **do**

        $\alpha_{i,R}^{a,s'} = \theta_a^{s,s'}\alpha_{i,R}$, $\alpha_{i,C}^{a,s'} = \theta_a^{s,s'}\alpha_{i,C}$    // multiplied by variable $\theta_a^{s,s'}$

// backup for each action

**foreach** $a \in A$ **do**

    Solve the following LP to obtain best randomized action at the next time step:

$$\max_{\tilde{w}_{is'}, d_z} \sum_{i,s'} \tilde{w}_{is'}\alpha_{i,R}^{a,s'}(b) \quad \text{subject to} \quad \begin{array}{l} \sum_i \tilde{w}_{is'}\alpha_{i,C}^{a,s'}(b) \leq d_{s'} \quad \forall s' \\ \sum_i \tilde{w}_{is'} = 1 \quad \forall s' \\ \tilde{w}_{is'} \geq 0 \quad \forall i, s' \\ \sum_z d_{s'} = \frac{1}{\gamma}(d - C(s,a)) \end{array}$$

    $\alpha_R^a = \alpha_R^{a,*} + \gamma \sum_{i,s'} \tilde{w}_{is'}\alpha_{i,R}^{a,s'}$, $\alpha_C^a = \alpha_C^{a,*} + \gamma \sum_{i,s'} \tilde{w}_{is'}\alpha_{i,C}^{a,s'}$

// find the best randomized action for the current time step

Solve the following LP with :

$$\max_{w_a} \sum_a w_a \alpha_R^a(b) \quad \text{subject to} \quad \begin{array}{l} \sum_a w_a \alpha_C^a(b) \leq d \\ \sum_a w_a = 1 \\ w_a \geq 0 \quad \forall a \end{array}$$

**return** $\Gamma'_{(s,n,d)} = \{(\alpha_R^a, \alpha_C^a) | w_a > 0\}$

---

We call our algorithm CBEETLE, which solves the hyperstate CPOMDP planning problem. As in BEETLE, $\alpha$-vectors for the expected total reward and cost are represented as $\alpha$-functions in terms of unknown parameters. The point-based backup operator in Algorithm 1 naturally extends to $\alpha$-functions without significant increase in the computation complexity: the size of LP does not increase even though the belief states represent probability distributions over unknown parameters. Algorithm 2 shows the point-based backup of $\alpha$-functions in the hyperstate CPOMDP. In addition, if we choose a fixed set of basis functions for representing $\alpha$-functions, we can pre-compute the projections of $\alpha$-functions ($\tilde{T}$, $\tilde{R}$, and $\tilde{C}$) in the same way as BEETLE. This technique is used in the point-based backup, although not explicitly described in the pseudocode due to the page limit.

We also implemented the randomized point-based backup to further improve the performance. The key step in the randomized value update is to check whether a newly generated $\alpha$-function pairs $\Gamma = \{(\alpha_{i,R}, \alpha_{i,C})\}$ from a point-based backup yields improved value at some other sampled belief state $(s, n, d)$. We can obtain the value of $\Gamma$ at the belief state by solving the following LP:

$$\max_{w_i} \sum_i w_i \alpha_{i,R}(b) \quad \text{subject to} \quad \begin{array}{l} \sum_i w_i \alpha_{i,C}(b) \leq d \\ \sum_i w_i = 1 \\ w_i \geq 0 \quad \forall i \end{array} \tag{2}$$

If we can find an improved value, we skip the point-based backup at $(s, n, d)$ in the current iteration. Algorithm 3 shows the randomized point-based value update.

In summary, the point-based value iteration algorithm for CPOMDP and BEETLE readily provide all the essential computational tools to implement the hyperstate CPOMDP planning for the cost-sensitive BRL.

**Algorithm 3:** Randomized point-based value update for the hyperstate CPOMDP

---

**input** : set $B$ of sampled belief points, and set $\Gamma_s$ of $\alpha$-function pairs for each state $s$
**output**: set $\Gamma'_s$ of $\alpha$-function pairs (updated value function)
// initialize
$\tilde{B} = B$  // belief points needed to be improved
**foreach** $s \in S$ **do**
    $\Gamma'_s = \emptyset$
// randomized backup
**while** $\tilde{B} \neq \emptyset$ **do**
    Sample $\tilde{b} = (\tilde{s}, \tilde{n}, \tilde{d}) \in \tilde{B}$
    Obtain $\Gamma'_{\tilde{b}}$ by point-based backup at $\tilde{b}$ with $\{\Gamma_s | \forall s \in S\}$ (Algorithm 2)
    $\Gamma'_{\tilde{s}} = \Gamma'_{\tilde{s}} \cup \Gamma'_{\tilde{b}}$
    **foreach** $b \in B$ **do**
        Calculate $V'(b)$ by solving the LP Eqn. 2 with $\Gamma'_{\tilde{b}}$
    $\tilde{B} = \{b \in B : V'(b) < V(b)\}$
**return** $\{\Gamma'_s | \forall s \in S\}$

---

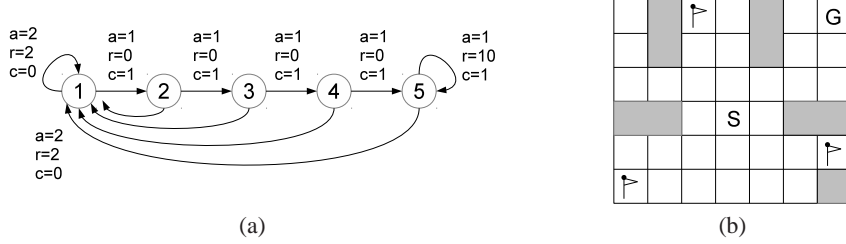

(a)                                       (b)

Figure 1: (a) 5-state chain: each edge is labeled with action, reward, and cost associated with the transition. (b) $6 \times 7$ maze: a $6 \times 7$ grid including the start location with recharging station (S), goal location (G), and 3 flags to capture.

## 4   Experiments

We used the constrained versions of two standard BRL problems to demonstrate the cost-sensitive exploration. The first one is the 5-state chain [15, 16, 4], and the second one is the $6 \times 7$ maze [16].

### 4.1   Description of Problems

The 5-state chain problem is shown in Figure 1a, where the agent has two actions 1 and 2. The agent receives a large reward of 10 by executing action 1 in state 5, or a small reward of 2 by executing action 2 in any state. With probability 0.2, the agent slips and makes the transition corresponding to the other action. We defined the constrained version of the problem by assigning a cost of 1 for action 1 in every state, thus making the consecutive execution of action 1 potentially violate the cost constraint.

The $6 \times 7$ maze problem is shown in Figure 1b, where the white cells are navigatable locations and gray cells are walls that block navigation. There are 5 actions available to the agent: move left, right, up, down, or stay. Every "move" action (except for the stay action) can fail with probability 0.1, resulting in a slip to two nearby cells that are perpendicular to the intended direction. If the agent bumps into a wall, the action will have no effect. The goal of this problem is to capture as many flags as possible and reach the goal location. Upon reaching the goal, the agent obtains a reward equal to the number of flags captured, and the agent gets warped back to the start location. Since there are 33 reachable locations in the maze and 8 possible combinations for the status of captured flags, there are a total of 264 states. We defined the constrained version of the problem by assuming that the agent is equipped with a battery and every action consumes energy except the stay action at

recharging station. We modeled the power consumption by assigning a cost of 0 for executing the stay action at the recharging station, and a cost of 1 otherwise. Thus, the battery recharging is done by executing stay action at the recharging station, as the admissible cost increases by factor $1/\gamma$.[3]

## 4.2 Results

Table 1 summarizes the experimental results for the constrained chain and maze problems.

In the chain problem, we used two structural prior models, "tied" and "semi", among three priors experimented in [4]. Both chain-tied and chain-semi assume that the transition dynamics are known to the agent except for the slip probabilities. In chain-tied, the slip probability is assumed to be independent of state and action, thus there is only one unknown parameter in transition dynamics. In chain-semi, the slip probability is assumed to be action dependent, thus there are two unknown parameters since there are two actions. We used uninformative Dirichlet priors in both settings. We excluded experimenting with the "full" prior model (completely unknown transition dynamics) since even BEETLE was not able to learn a near-optimal policy as reported in [4].

We report the average discounted total reward and cost as well as their 95% confidence intervals for the first 1000 time steps using 200 simulated trials. We performed 60 Bellman iterations on 500 belief states, and used the first 50 belief states for choosing the set of basis functions. The discount factor was set to 0.99.

When $\hat{c}$=100, which is the maximum expected total cost that can be incurred by any policy, CBEE-TLE found policies that are as good as the policy found by BEETLE since the cost constraint has no effect. As we impose tighter cost constraints by $\hat{c}$=75, 50, and 25, the policies start to trade off the rewards in order to meet the cost constraint. Note also that, although we use approximations in the various stages of the algorithm, $\hat{c}$ is within the confidence intervals of the average total cost, meaning that the cost constraint is either met or violated by statistically insignificant amounts. Since chain-semi has more unknown parameters than chain-tied, it is natural that the performance of CBEETLE policy is slighly degraded in chain-semi. Note also that as we impose tighter cost constraints, the running times generally increase. This is because the cost constraint in the LP tends to become active at more belief states, generating two $\alpha$-function pairs instead of a single $\alpha$-function pair when the cost constaint in the LP is not active.

The results for the maze problem were calculated for the first 2000 time steps using 100 simulated trials. We performed 30 Bellman iterations on 2000 belief states, and used 50 basis functions. Due to the computational requirement for solving the large hyperstate CPOMDP, we only experimented with the "tied" prior model which assumes that the slip probability is shared by every state and action. Running CBEETLE with $\hat{c} = 1/(1 - 0.95) = 20$ is equivalent to running BEETLE without cost constraints, as verified in the table.

We further analyzed the cost-sensitive exploration behaviour in the maze problem. Figure 2 compares the policy behaviors of BEETLE and CBEETLE($\hat{c}$=18) in the maze problem. The BEETLE policy generally captures the top flag first (Figure 2a), then navigates straight to the goal (Figure 2b) or captures the right flag and navigates to the goal (Figure 2c). If it captures the right flag first, it then navigates to the goal (Figure 2d) or captures the top flag and navigates to the goal (Figure 2e). We suspect that the reason the third flag on the left is not captured is due to the relatively low discount rate, hence ignored due to numerical approximations. The CBEETLE policy shows a similar capture behaviour, but it stays at the recharging station for a number of time steps between the first and second flag captures, which can be confirmed by the high state visitation frequency for the cell S in Figures 2g and 2i. This is because the policy cannot navigate to the other flag position and move to the goal without recharging the battery in between. The agent also frequently visits the recharging station before the first flag capture (Figure 2f) because it actively explores for the first flag with a high uncertainty in the dynamics.

Table 1: Experimental results for the chain and maze problems.

| problem | algorithm | $\hat{c}$ | utopic value | avg discounted total reward | avg discounted total cost | time (minutes) |
|---|---|---|---|---|---|---|
| chain-tied $|S| = 5$ $|A| = 2$ | BEETLE | — | 354.77 | 351.11±8.42 | — | 1.0 |
| | CBEETLE | 100 | 354.77 | 354.68±8.57 | 100.00±0 | 2.4 |
| | | 75 | 325.75 | 287.70±8.17 | 75.05±0.14 | 2.4 |
| | | 50 | 296.73 | 264.97±7.06 | 49.96±0.09 | 44.3 |
| | | 25 | 238.95 | 212.19±4.98 | 25.12±0.13 | 80.59 |
| chain-semi $|S| = 5$ $|A| = 2$ | BEETLE | — | 354.77 | 351.11±8.42 | — | 1.6 |
| | CBEETLE | 100 | 354.77 | 354.68±8.57 | 100.00±0 | 3.7 |
| | | 75 | 325.75 | 287.64±8.16 | 75.05±0.14 | 3.8 |
| | | 50 | 296.73 | 256.76±7.23 | 50.09±0.14 | 70.7 |
| | | 25 | 238.95 | 204.84±4.51 | 25.01±0.16 | 139.3 |
| maze-tied $|S| = 264$ $|A| = 5$ | BEETLE | — | 1.03 | 1.02±0.02 | — | 159.8 |
| | CBEETLE | 20 | 1.03 | 1.02±0.02 | 19.04±0.02 | 242.5 |
| | | 18 | 0.97 | 0.93±0.04 | 17.96±0.46 | 733.1 |

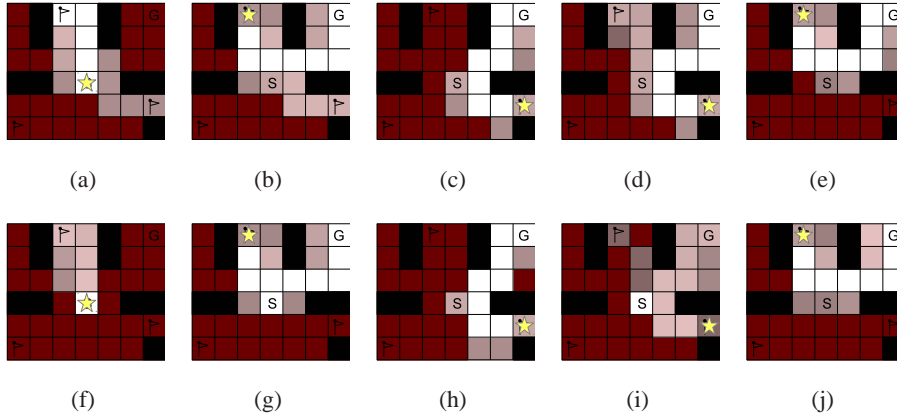

| (a) | (b) | (c) | (d) | (e) |
|---|---|---|---|---|

| (f) | (g) | (h) | (i) | (j) |
|---|---|---|---|---|

Figure 2: State visitation frequencies of each location in the maze problem over 100 runs. Brightness is proportional to the relative visitation frequency. (a-e) Behavior of BEETLE (a) before the first flag capture, (b) after the top flag captured first, (c) after the top flag captured first and the right flag second, (d) after the right flag captured first, and (e) after the right flag captured first and the top flag second. (f-j) Behavior of CBEETLE ($\hat{c} = 18$). The yellow star represents the current location of the agent.

## 5  Conclusion

In this paper, we proposed CBEETLE, a model-based BRL algorithm for cost-sensitive exploration, extending BEETLE to solve the hyperstate CPOMDP which models BRL using cost constraints. We showed that cost-sensitive BRL can be effectively solved by the randomized point-based value iteration for CPOMDPs. Experimental results show that CBEETLE can learn reasonably good policies for underlying CMDPs while exploring the unknown environment cost-sensitively.

While our experiments show that the policies generally satisfy the cost constraints, it can still potentially violate the constraints since we approximate the alpha functions using a finite number of basis functions. As for the future work, we plan to focus on making CBEETLE more robust to the approximation errors by performing a constrained optimization when approximating alpha functions to guarantee that we never violate the cost constraints.

## Acknowledgments

This work was supported by National Research Foundation of Korea (Grant# 2012-007881), the Defense Acquisition Program Administration and Agency for Defense Development of Korea (Contract# UD080042AD), and the SW Computing R&D Program of KEIT (2011-10041313) funded by the Ministry of Knowledge Economy of Korea.

## Footnotes

[1]Note that this algorithm is an improvement over the heuristic distribution of the admissible cost to each observation by ratio $\Pr(z|b, a)$ in [12]. Instead, we optimize the cost distribution by solving an LP.

[2]The $\alpha$-functions in the pseudocode are functions of $\theta$ and $\alpha(b)$ is defined to be $\int_\theta b(\theta)\alpha(\theta)d\theta$ as explained in Sec. 2.2.

[3]It may seem odd that the battery recharges at an exponential rate. We can set $\gamma = 1$ and make the cost function assign, e.g., a cost of -1 for recharging and 1 for consuming, but our implementation currently assumes same discount factor for the rewards and costs. Implementation for different discount factors is left as a future work, but note that we can still obtain meaningful results with $\gamma$ sufficiently close to 1.

# References

[1] R. Howard. *Dynamic programming*. MIT Press, 1960.

[2] M. Duff. *Optimal learning: Computational procedures for Bayes-adaptive Markov decision processes*. PhD thesis, University of Massachusetts, Amherst, 2002.

[3] S. Ross, J. Pineau, B. Chaib-draa, and P. Kreitmann. A Bayesian approach for larning and planning in partially observable markov decision processes. *Journal of Machine Learning Research*, 12, 2011.

[4] P. Poupart, N. Vlassis, J. Hoey, and K. Regan. An analytic solution to descrete Bayesian reinforcement learning. In *Proc. of ICML*, 2006.

[5] E. Altman. *Constrained Markov Decision Processes*. Chapman & Hall/CRC, 1999.

[6] K. W. Ross and R. Varadarajan. Markov decision-processes with sample path constraints - the communicating case. *Operations Research*, 37(5):780–790, 1989.

[7] K. W. Ross and R. Varadarajan. Multichain Markov decision-processes with a sample path constraint - a decomposition approach. *Mathematics of Operations Research*, 16(1):195–207, 1991.

[8] E. Delage and S. Mannor. Percentile optimization for Markov decision processes with parameter uncertainty. *Operations Research*, 58(1), 2010.

[9] A. Hans, D. Schneegaß, A. M. Schäfer, and S. Udluft. Safe exploration for reinforcement learning. In *Proc. of 16th European Symposium on Artificial Neural Networks*, 2008.

[10] T. M. Moldovan and P. Abbeel. Safe exploration in Markov decision processes. In *Proc. of NIPS Workshop on Bayesian Optimization, Experimental Design and Bandits*, 2011.

[11] J. D. Isom, S. P. Meyn, and R. D. Braatz. Piecewise linear dynamic programming for constrained POMDPs. In *Proc. of AAAI*, 2008.

[12] D. Kim, J. Lee, K.-E. Kim, and P. Poupart. Point-based value iteration for constrained POMDPs. In *Proc. of IJCAI*, 2011.

[13] A. B. Piunovskiy and X. Mao. Constrained Markovian decision processes: the dynamic programming approach. *Operations Research Letters*, 27(3):119–126, 2000.

[14] M. T. J. Spaan and N. Vlassis. Perseus: Randomized point-based value iteration for POMDPs. *Journal of Artificial Intelligence Research*, 24, 2005.

[15] R. Dearden, N. Friedman, and D. Andre. Bayesian Q-learning. In *Proc. of AAAI*, 1998.

[16] M. Strens. A Bayesian framework for reinforcement learning. In *Proc. of ICML*, 2000.

